# Unsupervised Parallel Feature Extraction from First Principles

**Mats Österberg**
Image Processing Laboratory
Dept. EE., Linköping University
S-58183 Linköping Sweden

**Reiner Lenz**
Image Processing Laboratory
Dept. EE., Linköping University
S-58183 Linköping Sweden

## Abstract

We describe a number of learning rules that can be used to train unsupervised parallel feature extraction systems. The learning rules are derived using gradient ascent of a quality function. We consider a number of quality functions that are rational functions of higher order moments of the extracted feature values. We show that one system learns the principle components of the correlation matrix. Principal component analysis systems are usually not optimal feature extractors for classification. Therefore we design quality functions which produce feature vectors that support unsupervised classification. The properties of the different systems are compared with the help of different artificially designed datasets and a database consisting of all Munsell color spectra.

## 1 Introduction

There are a number of unsupervised Hebbian learning algorithms (see Oja, 1992 and references therein) that perform some version of the Karhunen-Loéve expansion. Our approach to unsupervised feature extraction is to identify some desirable properties of the extracted feature vectors and to construct a quality functions that measures these properties. The filter functions are then learned from the input patterns by optimizing this selected quality function. In comparison to conventional unsupervised Hebbian learning this approach reduces the amount of communication between the units needed to learn the weights in parallel since the complexity now lies in the learning rule used.

The optimal (orthogonal) solution to two of the proposed quality functions turn out to be related to the Karhunen-Loéve expansion: the first learns an arbitrary rotation of the eigenvectors whereas the later learns the pure eigenvectors. A common problem with the Karhunen-Loéve expansion is the fact that the first eigenvector is normally the mean vector of the input patterns. In this case one filter function will have a more or less uniform response for a wide range of input patterns which makes it rather useless for classification. We will show that one quality function leads to a system that tend to learn filter functions which have a large magnitude response for just one class of samples (different for each filter function) and low magnitude response for samples from all other classes. Thus, it is possible to classify an incoming pattern by simply observing which filter function has the largest magnitude response. Similar to Intrator's Projection Pursuit related network (see Intrator & Cooper, 1992 and references therein) some quality functions use higher order ($> 2$) statistics of the input process but in contrast to Intrator's network there is no need to specify the amount of lateral inhibition needed to learn different filter functions.

All systems considered in this paper are linear but at the end we will briefly discuss possible non-linear extensions.

## 2   Quality functions

In the following we consider linear filter systems. These can be described by the equation:

$$O(t) \quad = \quad W(t)P(t) \tag{1}$$

where $P(t) \in \mathbf{R}^{M:1}$ is the input pattern at iteration t, $W(t) \in \mathbf{R}^{N:M}$ is the filter coefficient matrix and $O(t) = (o_1(t), \ldots, o_N(t))' \in \mathbf{R}^{N:1}$ is the extracted feature vector. Usually $M > N$, i.e. the feature extraction process defines a reduction of the dimensionality. Furthermore, we assume that both the input patterns and the filter functions are normed; $||P(t)|| = 1$ and $||W_n(t)|| = 1$, $\forall t\, \forall n$. This implies that $|o_n^2(t)| \leq 1, \forall t\, \forall n$.

Our first decision is to measure the scatter of the extracted feature vectors around the origin by the determinant of the output correlation matrix:

$$Q_{MS}(t) = \det E_t\{O(t)O'(t)\} \tag{2}$$

$Q_{MS}(t)$ is the quality function used in the Maximum Scatter Filter System (MS-system). The use of the determinant is motivated by the following two observations: 1. The determinant is equal to the product of the eigenvalues and hence the product of the variances in the principal directions and thus a measure of the scattering volume in the feature space. 2. The determinant vanishs if some filter functions are linearly dependent.

In (Lenz & Österberg, 1992) we have shown that the optimal filter functions to $Q_{MS}(t)$ are given by an arbitrary rotation of the N eigenvectors corresponding to the N largest eigenvalues of the input correlation matrix:

$$W_{opt} \quad = \quad RU_{eig} \tag{3}$$

where $U_{eig}$ contains the largest eigenvectors (or principal components) of the input correlation matrix $E_t\{P(t)P'(t)\}$. $R$ is an arbitrary rotation matrix with $\det(R) = 1$. To differentiate between these solutions we need a second criterion.

One attempt to define the best rotation is to require that the mean energy $E_t\{o_n^2(t)\}$ should be concentrated in as few components $o_n(t)$ of the extracted feature vector as possible. Thus, the mean energy $E_t\{o_n^2(t)\}$ of each filter function should be either very high (i.e. near 1) or very low (i.e. near 0). This leads to the following second order concentration measure:

$$Q_2(t) = \sum_{n=1}^{N} E_t\{o_n^2(t)\}\left(1 - E_t\{o_n^2(t)\}\right) \tag{4}$$

which has a low non-negative value if the energies are concentrated.

Another idea is to find a system that produces feature vectors that have unsupervised discrimination power. In this case each learned filter function should respond selectively, i.e. have a large response for some input samples and low response for others. One formulation of this goal is that each extracted feature vector should be (up to the sign) binary; $o_i(t) = \pm 1$ and $o_n(t) = 0, n \neq i, \forall t$. This can be measured by the following fourth order expression:

$$Q_4(t) = E_t\{\sum_{n=1}^{N} o_n^2(t)\left(1 - o_n^2(t)\right)\} = \sum_{n=1}^{N} E_t\{o_n^2(t)\} - E_t\{o_n^4(t)\} \tag{5}$$

which has a low non-negative value if the features are binary. Note that it is not sufficient to use $o_n(t)$ instead of $o_n^2(t)$ since $Q_4(t)$ will have a low value also for feature vectors with components equal in magnitude but with opposite sign. A third criterion can be found as follows: if the filter functions have selective filter response then the response to different input patterns differ in magnitude and thus the variance of the mean energy $E_t\{o_n^2(t)\}$ is large. The total variance is measured by:

$$Q_{\text{Var}}(t) = \sum_{n=1}^{N} \text{Var}\{o_n^2(t)\} = \sum_{n=1}^{N} E_t\{(o_n^2(t) - E_t\{o_n^2(t)\})^2\}$$

$$= \sum_{n=1}^{N} E_t\{o_n^4(t)\} - \left(E_t\{o_n^2(t)\}\right)^2 \tag{6}$$

Following (Darlington, 1970) it can be shown that the distribution of $o_n^2$ should be bimodal (modes below and above $E_t\{o_n^2\}$) to maximize $Q_{\text{Var}}(t)$. The main difference between $Q_{\text{Var}}(t)$ and the quality function used by Intrator is the use of a fourth order term $E_t\{o_n^4(t)\}$ instead of a third order term $E_t\{o_n^3(t)\}$. With $E_t\{o_n^3(t)\}$ the quality function is a measure of the skewness of the distribution $o(t)$ and it is maximized when one mode is at zero and one (or several) is above $E_t\{o_n^2(t)\}$.

In this paper we will examine the following non-parametric combinations of the quality functions above:

$$Q_{KL}(t) = \frac{Q_{MS}(t)}{Q_2(t)} \tag{7}$$

$$Q_{FO}(t) = \frac{Q_{MS}(t)}{Q_4(t)} \tag{8}$$

$$Q_{MV}(t) = Q_{\text{Var}}(t)Q_{MS}(t) \tag{9}$$

We refer to the corresponding filter systems as: the Karhunen-Loéve Filter System (KL-system), the Fourth Order Filter System (FO-system) and the Maximum Variance Filter System (MV-system).

Since each quality function is a combination of two different functions it is hard to find the global optimal solution. Instead we use the following strategy to determine a local optimal solution.

**Definition 1** *The optimal orthogonal solution to each quality function is of the form:*

$$W_{opt} \quad = \quad R_{opt}U_{eig} \tag{10}$$

*where $R_{opt}$ is the rotation of the largest eigenvectors which minimize $Q_2(t)$, $Q_4(t)$ or maximize $Q_{\mathrm{Var}}(t)$.*

In (Lenz & Österberg, 1992 and Österberg, 1993) we have shown that the optimal orthogonal solution to the KL-system are the N pure eigenvectors if the N largest eigenvalues are all distinct (i.e. $R_{opt} = I$). If some eigenvalues are equal then the solution is only determined up to an arbitrary rotation of the eigenvectors with equal eigenvalues. The fourth order term $E_t\{o_n^4(t)\}$ in $Q_4(t)$ and $Q_{\mathrm{Var}}(t)$ makes it difficult to derive a closed form solution. The best we can achieve is a numerical method (in the case of $Q_4(t)$ see Österberg, 1993) for the computation of the optimal orthogonal filter functions.

## 3  Maximization of the quality function

The partial derivatives of $Q_{MS}(t)$, $Q_2(t)$, $Q_4(t)$ and $Q_{\mathrm{Var}}(t)$ with respect to $w_n^m(t)$ (the $m^{th}$ weight in the $n^{th}$ filter function at iteration $t$) are only functions of the input pattern $P(t)$, the output values $O(t) = (o_1(t), \ldots, o_N(t))$ and the previous values of the weight coefficients $(w_n^1(t-1), \ldots, w_n^M(t-1))$ within the filter function (see Österberg, 1993). Especially, they are not functions of the internal weights $((w_i^1(t-1), \ldots, w_i^M(t-1)), i \neq n)$ of the other filter functions in the system. This implies that the filter coefficients can be learned in parallel using a system of the structure shown in Figure 1.

In (Österberg, 1993) we used on-line optimization techniques based on gradient ascent. We tried two different methods to select the step length parameter. One rather heuristical depending on the output $o_n(t)$ of the filter function and one inverse proportional to the second partial derivative of the quality function with respect to $w_n^m(t)$. In each iteration the length of each filter function was explicitly normalized to one. Currently, we investigate standard unconstrained optimization methods (Dennis & Schnabel, 1983) based on batch learning. Now the step length parameter is selected by line search in the search direction $S(t)$:

$$\max_{\lambda} Q(W(t) + \lambda S(t)) \tag{11}$$

Typical choices of $S(t)$ include $S(t) = I$ and $S(t) = H^{-1}$. With the identity matrix we get Steepest Ascent and with the inverse Hessian the quasi-Newton algorithm. Using sufficient synchronism the line search can be incorporated in the parallel structure (Figure 1). To incorporate the quasi-Newton algorithm we have to assume

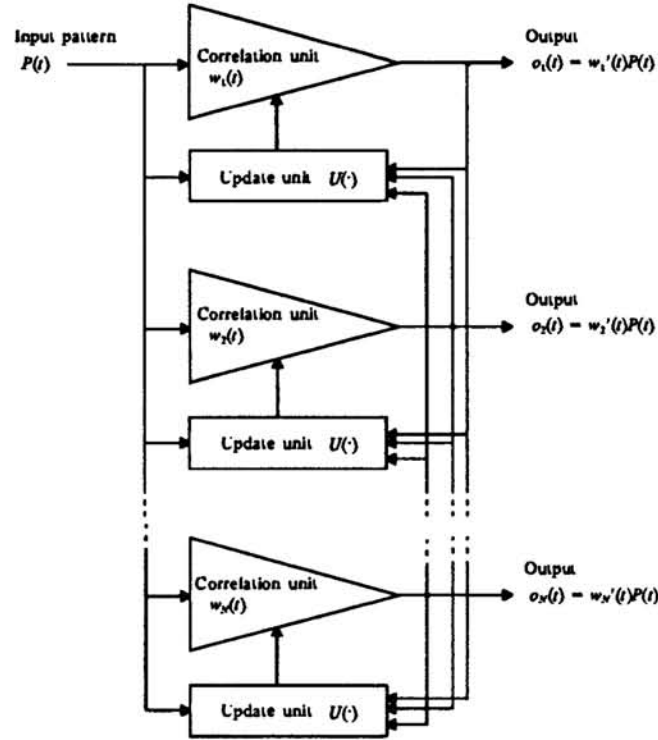

Figure 1: The architecture of the filter system

that the Hessian matrix is block diagonal, i.e. the second partial derivatives with respect to $w_k^m(t)w_l^m(t), k \neq l, \forall m$ are assumed to be zero. In general this is not the case and it is not clear if a block diagonal approximation is valid or not. The second partial derivatives can be approximated by secant methods (normally the BFGS method). Furthermore the condition of normalized filter functions can be achieved by optimizing in hyperspherical polar coordinates. Preliminary experiments (mustly with Steepest Ascent) show that more advanced optimization techniques lead to a more robust convergence of the filter functions.

## 4    Experiments

In (Österberg, 1993) we describe a series of experiments in which we investigate systematically the following properties of the MS-system, the KL-system and the FO-system: convergence speed, dependence on initial solution $W(0)$ , distance between learned solution and optimal (orthogonal) solution, supervised classification of the extracted feature vectors using linear regression and the degree of selective response of the learned filter functions. We use training sets with controlled scalar products between the cluster centers of three classes of input patterns embedded in a 32-D space. The results of the experiments can be summarized as follows. In contrast to the MS-system, we noticed that the KL- and FO-system had problems to converge to the optimal orthogonal solutions for some initial solutions. All systems learned orthogonal solutions regardless of $W(0)$. The supervised classification power was independent of the filter system used. Only the FO-system produced

Table 1: Typical filter response to patterns from (a)-(c) Tset$_1$ and (d) Tset$_2$ using the filter functions learned with (a) the KL-system, (b) the FO-system and (c)-(d) the MV-system. (e)-(f) Output covariance matrix using the filter functions learned with (e) the KL-system and (f) the MV-system.

$$\left[\begin{pmatrix} -0.12 \\ 0.92 \\ -0.38 \end{pmatrix}, \begin{pmatrix} -0.46 \\ 0.83 \\ 0.32 \end{pmatrix}, \begin{pmatrix} 0.73 \\ 0.66 \\ 0.14 \end{pmatrix}\right] \quad \left[\begin{pmatrix} -0.71 \\ 0.59 \\ 0.28 \end{pmatrix}, \begin{pmatrix} -0.99 \\ -0.08 \\ 0.01 \end{pmatrix}, \begin{pmatrix} -0.22 \\ -0.04 \\ 0.97 \end{pmatrix}\right]$$
$$\text{(a)} \qquad\qquad\qquad\qquad \text{(b)}$$

$$\left[\begin{pmatrix} 0.28 \\ -0.91 \\ 0.44 \end{pmatrix}, \begin{pmatrix} 0.10 \\ -0.39 \\ 0.95 \end{pmatrix}, \begin{pmatrix} 0.98 \\ -0.23 \\ 0.11 \end{pmatrix}\right] \quad \left[\begin{pmatrix} -0.50 \\ -0.80 \\ 0.50 \end{pmatrix}, \begin{pmatrix} -0.49 \\ -0.50 \\ 0.81 \end{pmatrix}, \begin{pmatrix} -0.81 \\ -0.49 \\ 0.50 \end{pmatrix}\right]$$
$$\text{(c)} \qquad\qquad\qquad\qquad \text{(d)}$$

$$\begin{pmatrix} 0.0340 & 0.0001 & 0.0005 \\ 0.0001 & 0.9300 & 0.0000 \\ 0.0005 & 0.0000 & 0.0353 \end{pmatrix} \qquad \begin{pmatrix} 0.3788 & 0.3463 & -0.3473 \\ 0.3463 & 0.3760 & -0.3467 \\ -0.3473 & -0.3467 & 0.3814 \end{pmatrix}$$
$$\text{(e)} \qquad\qquad\qquad\qquad \text{(f)}$$

filter functions which mainly react for patterns from just one class and only if the similarity (measured by the scalar product) between the classes in the training set was smaller than approximately 0.5. Thus, the FO-system extracts feature vectors which have unsupervised discrimination power. Furthermore, we showed that the FO-system can distinguish between data sets having identical correlation matrices (second order statistics) but different fourth order statistics. Recent experiments with more advanced optimization techniques (Steepest Ascent) show better convergence properties for the KL- and FO-system. Especially the distance between the learned filter functions and the optimal orthogonal ones becomes smaller.

We will describe some experiments which show that the MV-system is more suitable for tasks requiring unsupervised classification. We use two training sets Tset$_1$ and Tset$_2$. In the first set the mean scalar product between class one and two is 0.7, between class one and three 0.5 and between class two and three 0.3. In the second set the mean scalar products between all classes are 0.9, i.e. the angle between all cluster centers is arccos(0.9) = 26°. In Table 4(a)-(c) we show the filter response of the learned filter functions with the KL-, FO- and MV-system to typical examples of the input patterns in the training set Tset$_1$. For the KL-system we see that the second filter function gives the largest magnitude response for both, patterns from class one and two. For the FO-system the feature vectors are more binary. Still the first filter function has the largest magnitude response for patterns from class one and two. For the MV-system we see that each filter function has largest magnitude response for only one class of input patterns and thus the extracted feature vectors support unsupervised discrimination. In Table 4(d) (computed from Tset$_2$) we see that this is the case even then the scalar products between the cluster centers are as high as 0.9. The filter functions learned by the MV-system are approximately orthogonal. The system learns thus the rotation of the largest eigenvectors which maximizes $Q_{\mathrm{Var}}(t)$. Therefore it will not extract uncorrelated features (see Ta-

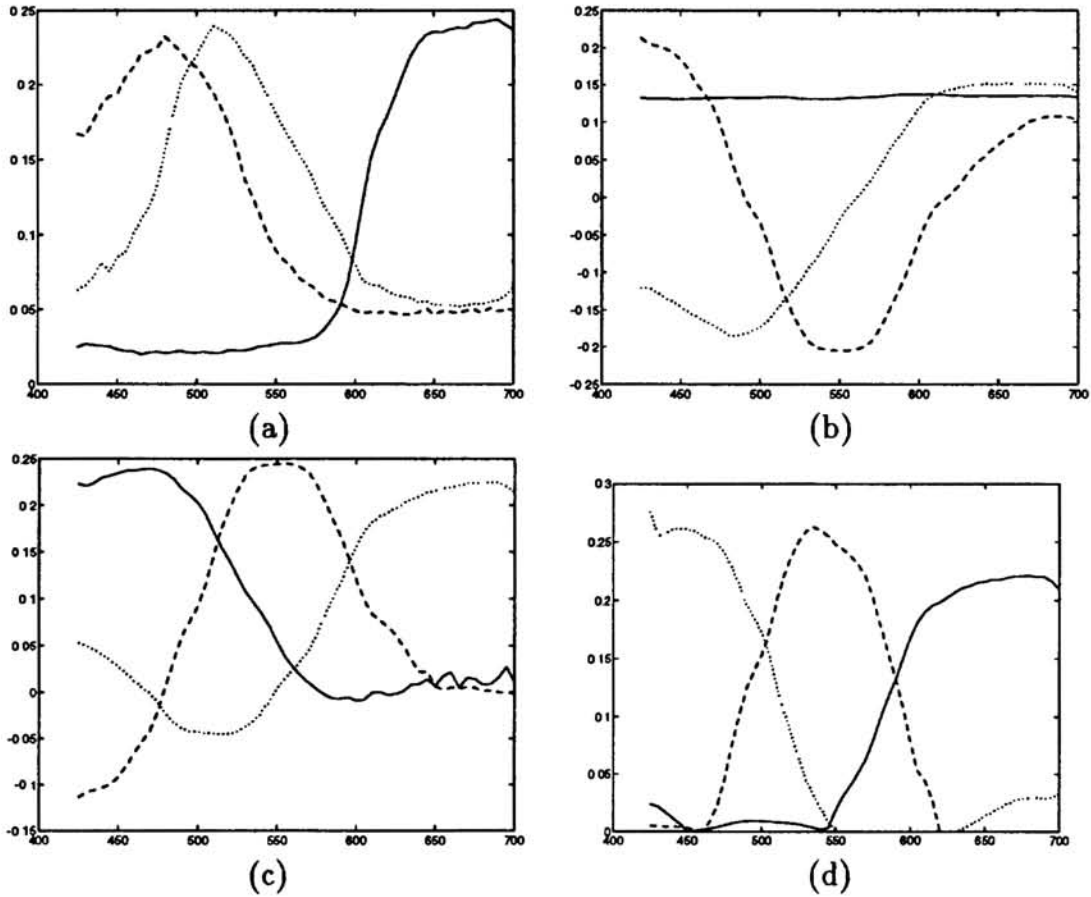

Figure 2: (a) Examples of normalized reflectance spectra of typical reddish (solid curve), greenish (dotted curve) and bluish (dashed curve) Munsell color chips. (b) The three largest eigenvectors belonging to the correlation matrix of the 1253 different reflectance spectra. (c) The learned filter functions with the MV-system. (d) The learned non-negative filter functions with the MV-system. In all figures the x-axes show the wave length (nm)

ble 4(f)) but the variances (e.g. the diagonal elements of the covariance matrix) of the features are more or less equal. In Table 4(e) we see that the KL-system extracts uncorrelate features with largely different variance. This demonstrates that the KL-system tries to learn the pure eigenvectors.

Recently, we have applied the MV-system to real world data. The training set consists of normalized reflectance spectra of the 1253 different color chips in the Munsell color atlas. Figure 2(a) shows one typical example of a red, a green and a blue color chip and Figure 2(b) the three largest eigenvectors belonging to the correlation matrix of the training set. We see that the first eigenvector (the solid curve) has a more or less uniform response for all different colors. On the other hand, the MV-system (Figure 2 (c)) learns one bluish, one greenish and one reddish filter function. Thus, the filter functions divide the color space according to the primary colors red, green and blue. We notice that the learned filter functions are orthogonal and tend to span the same space as the eigenvectors since $\|W_{sol} - R_{opt}U_{eig}\|_F = 0.0199$ (the Frobenius norm) where $R_{opt}$ maximizes $Q_{\mathrm{Var}}(t)$. Figure 2(d) show one preliminary attempt to include the condition of non-negative filter functions in the

optimization process (Steepest Ascent). We see that the learned filter functions are non-negative and divide the color space according to the primary colors. One possible real word application is optical color analysis where non-negative filter functions are much easier to realize using optical components. Smoother filter functions can be optained by incorporating additional constraints into the quality function.

# 5  Non-linear extensions

The proposed strategy to extract feature vectors apply to nonlinear filter systems as well. In this case the input output relation $O(t) = W(t)P(t)$ is replaced by $O(t) = f(W(t)P(t))$ where $f$ describes the desired non-linearity. The corresponding learning rule can be derived using gradient based techniques as long as the non-linearity $f(\cdot)$ is differentiable. The exact form of $f(\cdot)$ will usually be application oriented. Node nonlinearities of sigmoid type are one type of nonlinearities which has received a lot of attention (see for example Oja & Karhunen, 1993). Typical applications include: robust Principal Component Analysis PCA (outlier protection, noise suppression and symmetry breaking), sinusoidal signal detection in colored noise and robust curve fitting.

## Acknowledgements

This work was done under TFR-contract TFR-93-00192. The visit of M. Österberg at the Dept. of Info. Tech., Lappeenranta University of Technology was supported by a grant from the Nordic Research Network in Computer Vision. The Munsell color experiments were performed during this visit.

## References

R. B. Darlington. (1970) Is Kurtosis really peakedness? *American Statistics* **24**(2):19-20.

J. E. Dennis & Robert B. Schnabel. (1983) *Numerical Methods for Unconstrained Optimization and Nonlinear Equations.* Prentice-Hall.

N. Intrator & L.N. Cooper. (1992) Objective Function Formulation of the BCM Theory of Visual Cortical Plasticity: Statistical Connections, Stability Conditions. *Neural Networks* **5**:3-17.

R. Lenz & M. Österberg. (1992) Computing the Karhunen-Loeve expansion with a parallel, unsupervised filter system. *Neural Computations* **4**(3):382-392.

E. Oja. (1992) Principal Components, Minor Components, and Linear Neural Networks. *Neural Networks* **5**:927-935.

E. Oja & J. Karhunen. (1993) Nonlinear PCA: algorithms and Applications *Technical Report A18, Helsinki University of Technology, Laboratory of Computer and Information Sciences*, SF-02150 Espoo, Finland.

M. Österberg. (1993) Unsupervised Feature Extraction using Parallel Linear Filters. *Linköping Studies in Science and Technology. Thesis No. 372.*